# An Apobayesian Relative of Winnow

**Nick Littlestone**
NEC Research Institute
4 Independence Way
Princeton, NJ 08540

**Chris Mesterharm**
NEC Research Institute
4 Independence Way
Princeton, NJ 08540

## Abstract

We study a mistake-driven variant of an on-line Bayesian learning algorithm (similar to one studied by Cesa-Bianchi, Helmbold, and Panizza [CHP96]). This variant only updates its state (learns) on trials in which it makes a mistake. The algorithm makes binary classifications using a linear-threshold classifier and runs in time linear in the number of attributes seen by the learner. We have been able to show, theoretically and in simulations, that this algorithm performs well under assumptions quite different from those embodied in the prior of the original Bayesian algorithm. It can handle situations that we do not know how to handle in linear time with Bayesian algorithms. We expect our techniques to be useful in deriving and analyzing other apobayesian algorithms.

## 1  Introduction

We consider two styles of on-line learning. In both cases, learning proceeds in a sequence of trials. In each trial, a learner observes an *instance* to be classified, makes a *prediction* of its classification, and then observes a *label* that gives the correct classification. One style of on-line learning that we consider is Bayesian. The learner uses probabilistic assumptions about the world (embodied in a prior over some model class) and data observed in past trials to construct a probabilistic model (embodied in a posterior distribution over the model class). The learner uses this model to make a prediction in the current trial. When the learner is told the correct classification of the instance, the learner uses this information to update the model, generating a new posterior to be used in the next trial.

In the other style of learning that we consider, the attention is on the correctness of the predictions rather than on the model of the world. The internal state of the

learner is only changed when the learner makes a mistake (when the prediction fails to match the label). We call such an algorithm *mistake-driven*. (Such algorithms are often called *conservative* in the computational learning theory literature.) There is a simple way to derive a mistake-driven algorithm from any on-line learning algorithm (we restrict our attention in this paper to deterministic algorithms). The derived algorithm is just like the original algorithm, except that before every trial, it makes a record of its entire state, and after every trial in which its prediction is correct, it resets its state to match the recorded state, entirely forgetting the intervening trial. (Typically this is actually implemented not by making such a record, but by merely omitting the step that updates the state.) For example, if some algorithm keeps track of the number of trials it has seen, then the mistake-driven version of this algorithm will end up keeping track of the number of mistakes it has made. Whether the original or mistake-driven algorithm will do better depends on the task and on how the algorithms are evaluated.

We will start with a Bayesian learning algorithm that we call *SBSB* and use this procedure to derive a mistake-driven variant, *SASB*. Note that the variant cannot be expected to be a Bayesian learning algorithm (at least in the ordinary sense) since a Bayesian algorithm would make a prediction that minimizes the Bayes risk based on all the available data, and the mistake-driven variant has forgotten quite a bit. We call such algorithms *apobayesian* learning algorithms. This name is intended to suggest that they are derived from Bayesian learning algorithms, but are not themselves Bayesian. Our algorithm *SASB* is very close to an algorithm of [CHP96]. We study its application to different tasks than they do, analyzing its performance when it is applied to linearly separable data as described below.

In this paper instances will be chosen from the instance space $\mathbf{X} = \{0,1\}^n$ for some $n$. Thus instances are composed of $n$ boolean attributes. We consider only two category classifications tasks, with predictions and labels chosen from $\mathbf{Y} = \{0,1\}$.

We obtain a bound on the number of mistakes *SASB* makes that is comparable to bounds for various Winnow family algorithms given in [Lit88,Lit89]. As for those algorithms, the bound holds under the assumption that the points labeled 1 are linearly separable from the points labeled 0, and the bound depends on the size $\delta$ of the gap between the two classes. (See Section 3 for a definition of $\delta$.) The mistake bound for *SASB* is $O\left(\frac{1}{\delta^2} \log \frac{n}{\delta}\right)$. While this bound has an extra factor of $\log \frac{1}{\delta}$ not present in the bounds for the Winnow algorithms, SASB has the advantage of not needing any parameters. The Winnow family algorithms have parameters, and the algorithms' mistake bounds depend on setting the parameters to values that depend on $\delta$. (Often, the value of $\delta$ will not be known by the learner.) We expect the techniques used to obtain this bound to be useful in analyzing other apobayesian learning algorithms.

A number of authors have done related research regarding worst-case on-line loss bounds including [Fre96,KW95,Vov90]. Simulation experiments involving a Bayesian algorithm and a mistake-driven variant are described in [Lit95]. That paper provides useful background for this paper. Note that our present analysis techniques do not apply to the apobayesian algorithm studied there. The closest of the original Winnow family algorithms to *SASB* appears to be the Weighted Majority algorithm [LW94], which was analyzed for a case similar to that considered in this paper in [Lit89]. One should get a roughly correct impression of *SASB* if

one thinks of it as a version of the Weighted Majority algorithm that learns its parameters.

In the next section we describe the Bayesian algorithm that we start with. In Section 3 we discuss its mistake-driven apobayesian variant. Section 4 mentions some simulation experiments using these algorithms, and Section 5 is the conclusion.

## 2   A Bayesian Learning Algorithm

To describe the Bayesian learning algorithm we must specify a family of distributions over $\mathbf{X} \times \mathbf{Y}$ and a prior over this family of distributions. We parameterize the distributions with parameters $(\theta_1, \ldots, \theta_{n+1})$ chosen from $\mathbf{\Theta} = [0, 1]^{n+1}$. The parameter $\theta_{n+1}$ gives the probability that the label is 1, and the parameter $\theta_i$ gives the probability that the $i$th attribute matches the label. Note that the probability that the $i$th attribute is 1 given that the label is 1 equals the probability that the $i$th attribute is 0 given that the label is 0. We speak of this linkage between the probabilities for the two classes as a symmetry condition. With this linkage, the observation of a point from either class will affect the posterior distribution for both classes. It is perhaps more typical to choose priors that allow the two classes to be treated separately, so that the posterior for each class (giving the probability of elements of $\mathbf{X}$ conditioned on the label) depends only on the prior and on observations from that class. The symmetry condition that we impose appears to be important to the success of our analysis of the apobayesian variant of this algorithm. (Though we impose this condition to *derive* the algorithm, it turns out that the apobayesian variant can actually handle tasks where this condition is not satisfied.)

We choose a prior on $\mathbf{\Theta}$ that gives probability 1 to the set of all elements $\boldsymbol{\theta} = (\theta_1, \ldots, \theta_{n+1}) \in \mathbf{\Theta}$ for which at most one of $\theta_1, \ldots, \theta_n$ does not equal $\frac{1}{2}$. The prior is uniform on this set. Note that for any $\boldsymbol{\theta}$ in this set only a single attribute has a probability other than $\frac{1}{2}$ of matching the label, and thus only a single attribute is relevant. Concentrating on this set turns out to lead to an apobayesian algorithm that can, in fact, handle more than one relevant attribute and that performs particularly well when only a small fraction of the attributes are relevant.

This prior is related to to the familiar Naive Bayes model, which also assumes that the attributes are conditionally independent given the labels. However, in the typical Naive Bayes model there is no restriction to a single relevant attribute and the symmetry condition linking the two classes is not imposed.

Our prior leads to the following algorithm. (The name *SBSB* stands for "Symmetric Bayesian Algorithm with Singly-variant prior for Bernoulli distribution.")

**Algorithm** *SBSB*   Algorithm *SBSB* maintains counts $s_i$ of the number of times each attribute matches the label, a count $M$ of the number of times the label is 1, and a count $t$ of the number of trials.

**Initialization**   $s_i \leftarrow 0$ for $i = 1, \ldots, n$        $M \leftarrow 0$        $t \leftarrow 0$

**Prediction**   Predict 1 given instance $(x_1, \ldots, x_n)$ if and only if

$$(M + 1) \sum_{i=1}^{n} \frac{x_i(s_i+1)+(1-x_i)(t-s_i+1)}{\binom{t}{s_i}} > (t - M + 1) \sum_{i=1}^{n} \frac{(1-x_i)(s_i+1)+x_i(t-s_i+1)}{\binom{t}{s_i}}$$

**Update**   $M \leftarrow M + y$, $t \leftarrow t + 1$, and for each $i$, if $x_i = y$ then $s_i \leftarrow s_i + 1$

## 3 An Apobayesian Algorithm

We construct an apobayesian algorithm by converting algorithm *SBSB* into a mistake-driven algorithm using the standard conversion given in the introduction. We call the resulting learning algorithm *SASB*; we have replaced "Bayesian" with "Apobayesian" in the acronym.

In the previous section we made assumptions made about the generation of the instances and labels that led to *SBSB* and thence to *SASB*. These assumptions have served their purpose and we now abandon them. In analyzing the apobayesian algorithm we do not assume that the instances and labels are generated by some stochastic process. Instead we assume that the instance-label pairs in all of the trials are linearly-separable, that is, that there exist some $w_1, \ldots, w_n$, and $c$ such that for every instance-label pair $(\mathbf{x}, y)$ we have $\sum_{i=1}^{n} w_i x_i \geq c$ when $y = 1$ and $\sum_{i=1}^{n} w_i x_i \leq c$ when $y = 0$. We actually make a somewhat stronger assumption, given in the following theorem, which gives our bound for the apobayesian algorithm.

**Theorem 1** *Suppose that $\gamma_i \geq 0$ and $\overline{\gamma}_i \geq 0$ for $i = 1, \ldots, n$, and that $\sum_{i=1}^{n} \gamma_i + \overline{\gamma}_i = 1$. Suppose that $0 \leq b_0 < b_1 \leq 1$ and let $\delta = b_1 - b_0$. Suppose that algorithm SASB is run on a sequence of trials such that the instance $\mathbf{x}$ and label $y$ in each trial satisfy $\sum_{i=1}^{n} \gamma_i x_i + \overline{\gamma}_i (1 - x_i) \leq b_0$ if $y = 0$ and $\sum_{i=1}^{n} \gamma_i x_i + \overline{\gamma}_i (1 - x_i) \geq b_1$ if $y = 1$. Then the number of mistakes made by SASB will be bounded by $\frac{16}{\delta^2} \log \frac{8n}{\delta}$.*

We have space to say only a little about how the derivation of this bound proceeds. Details are given in [Lit96].

In analyzing *SASB* we work with an abstract description of the associated algorithm *SBSB*. This algorithm starts with a prior on $\Theta$ as described above. We represent this with a density $\rho_0$. Then after each trial it calculates a new posterior density $\rho_t(\boldsymbol{\theta}) = \frac{\rho_{t-1}(\boldsymbol{\theta}) P(\mathbf{x}, y | \boldsymbol{\theta})}{\int \rho_{t-1}(\boldsymbol{\theta}) P(\mathbf{x}, y | \boldsymbol{\theta})}$, where $\rho_t$ is the density after trial $t$ and $P(\mathbf{x}, y | \boldsymbol{\theta})$ is the conditional probability of the instance $\mathbf{x}$ and label $y$ observed in trial $t$ given $\boldsymbol{\theta}$. Thus we can think of the algorithm as maintaining a current distribution on $\Theta$ that is initially the prior. *SASB* is similar, but it leaves the current distribution unchanged when a mistake is not made. For there to exist a finite mistake bound there must exist some possible choice for the current distribution for which *SASB* would make perfect predictions, should it ever arrive at that distribution. We call any such distribution leading to perfect predictions a possible *target distribution*. It turns out that the separability condition given in Theorem 1 guarantees that a suitable target distribution exists. The analysis proceeds by showing that for an appropriate choice of a target density $\widetilde{\rho}$ the relative entropy of the current distribution with respect to the target distribution, $\int \widetilde{\rho}(\boldsymbol{\theta}) \log(\widetilde{\rho}(\boldsymbol{\theta})/\rho_t(\boldsymbol{\theta}))$, decreases by at least some amount $R > 0$ whenever a mistake is made. Since the relative entropy is never negative, the number of mistakes is bounded by the initial relative entropy divided by $R$. This form of analysis is very similar to the analysis of the various members of the Winnow family in [Lit89,Lit91].

The same technique can be applied to other apobayesian algorithms. The abstract update of $\rho_t$ given above is quite general. The success of the analysis depends on conditions on $\rho_0$ and $P(\mathbf{x}, y | \boldsymbol{\theta})$ that we do not have space here to discuss.

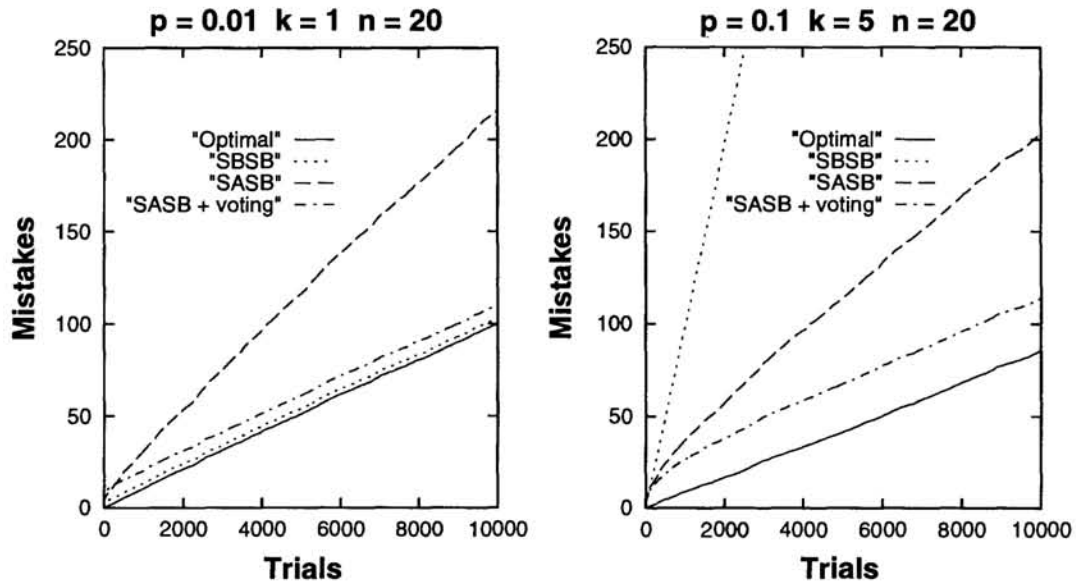

Figure 1: Comparison of *SASB* with *SBSB*

## 4  Simulation Experiments

The bound of the previous section was for perfectly linearly-separable data. We have also done some simulation experiments exploring the performance of *SASB* on non-separable data and comparing it with *SBSB* and with various other mistake-driven algorithms. A sample comparison of *SASB* with *SBSB* is shown in Figure 1. In each experimental run we generated 10000 trials with the instances and labels chosen randomly according to a distribution specified by $\theta_1 = \ldots = \theta_k = 1 - p$, $\theta_{k+1} = \ldots = \theta_{n+1} = .5$ where $\theta_1, \ldots, \theta_{n+1}$ are interpreted as specified in Section 2, $n$ is the number of attributes, and $n$, $p$, and $k$ are as specified at the top of each plot. The line labeled "optimal" shows the performance obtained by an optimal predictor that knows the distribution used to generate the data ahead of time, and thus does not need to do any learning. The lines labeled "SBSB" and "SASB" show the performance of the corresponding learning algorithms. The lines labeled "SASB + voting" show the performance of *SASB* with the addition of a voting procedure described in [Lit95]. This procedure improves the asymptotic mistake rate of the algorithms. Each line on the graph is the average of 30 runs. Each line plots the cumulative number of mistakes made by the algorithm from the beginning of the run as a function of the number of trials.

In the left hand plot, there is only 1 relevant attribute. This is exactly the case that *SBSB* is intended for, and it does better than *SASB*. In right hand plot, there are 5 relevant attributes; *SBSB* appears unable to take advantage of the extra information present in the extra relevant attributes, but *SASB* successfully does.

Comparison of *SASB* and previous Winnow family algorithms is still in progress, and we defer presenting details until a clearer picture has been obtained. *SASB* and the Weighted Majority algorithm often perform similarly in simulations. Typically, as one would expect, the Weighted Majority algorithm does somewhat better than

*SASB* when its parameters are chosen optimally for the particular learning task, and worse for bad choices of parameters.

## 5 Conclusion

Our mistake bounds and simulations suggest that *SASB* may be a useful alternative to the existing algorithms in the Winnow family. Based on the analysis style and the bounds, *SASB* should perhaps itself be considered a Winnow family algorithm. Further experiments are in progress comparing *SASB* with Winnow family algorithms run with a variety of parameter settings.

Perhaps of even greater interest is the potential application of our analytic techniques to a variety of other apobayesian algorithms (though as we have observed earlier, the techniques do not appear to apply to all such algorithms). We have already obtained some preliminary results regarding an interpretation of the Perceptron algorithm as an apobayesian algorithm. We are interested in looking for entirely new algorithms that can be derived in this way and also in better understanding the scope of applicability of our techniques. All of the analyses that we have looked at depend on symmetry conditions relating the probabilities for the two classes. It would be of interest to see what can be said when such symmetry conditions do not hold. In simulation experiments [Lit95], a mistake-driven variant of the standard Naive Bayes algorithm often does very well, despite the absence of such symmetry in the prior that it is based on.

Our simulation experiments and also the analysis of the related algorithm Winnow [Lit91] suggest that *SASB* can be expected to handle some instance-label pairs inside of the separating gap or on the wrong side, especially if they are not too far on the wrong side. In particular it appears to be able to handle data generated according to the distributions on which *SBSB* is based, which do not in general yield perfectly separable data.

It is of interest to compare the capabilities of the original Bayesian algorithm with the derived apobayesian algorithm. When the data is stochastically generated in a manner consistent with the assumptions behind the original algorithm, the original Bayesian algorithm can be expected to do better (see, for example, Figure 1). On the other hand, the apobayesian algorithm can handle data beyond the capabilities of the original Bayesian algorithm. For example, in the case we consider, the apobayesian algorithm can take advantage of the presence of more than one relevant attribute, even though the prior behind the original Bayesian algorithm assumes a single relevant attribute. Furthermore, as for all of the Winnow family algorithms, the mistake bound for the apobayesian algorithm does not depend on details of the behavior of the irrelevant attributes (including redundant attributes).

Instead of using the apobayesian variant, one might try to construct a Bayesian learning algorithm for a prior that reflects the actual dependencies among the attributes and the labels. However, it may not be clear what the appropriate prior is. It may be particularly unclear how to model the behavior of the irrelevant attributes. Furthermore, such a Bayesian algorithm may end up being computationally expensive. For example, attempting to keep track of correlations among all pairs of attributes may lead to an algorithm that needs time and space quadratic in the number of attributes. On the other hand, if we start with a Bayesian algorithm that

uses time and space linear in the number of attributes we can obtain an apobayesian algorithm that still uses linear time and space but that can handle situations beyond the capabilities of the original Bayesian algorithm.

**Acknowledgments**  This paper has benefited from discussions with Adam Grove.

# References

[CHP96] Nicolo Cesa-Bianchi, David P. Helmbold, and Sandra Panizza. On bayes methods for on-line boolean prediction. In *Proceedings of the Ninth Annual Conference on Computational Learning Theory*, pages 314–324, 1996.

[Fre96] Yoav Freund. Predicting a binary sequence almost as well as the optimal biased coin. In *Proceedings of the Ninth Annual Conference on Computational Learning Theory*, pages 89–98, 1996.

[KW95] J. Kivinen and M. K. Warmuth. Additive versus exponentiated gradient updates for linear prediction. In *Proc. 27th ACM Symp. on Theory of Computing*, pages 209–218, 1995.

[Lit88] N. Littlestone. Learning quickly when irrelevant attributes abound: A new linear-threshold algorithm. *Machine Learning*, 2:285–318, 1988.

[Lit89] N. Littlestone. *Mistake Bounds and Logarithmic Linear-threshold Learning Algorithms*. PhD thesis, Tech. Rept. UCSC-CRL-89-11, Univ. of Calif., Santa Cruz, 1989.

[Lit91] N. Littlestone. Redundant noisy attributes, attribute errors, and linear-threshold learning using Winnow. In *Proc. 4th Annu. Workshop on Comput. Learning Theory*, pages 147–156. Morgan Kaufmann, San Mateo, CA, 1991.

[Lit95] N. Littlestone. Comparing several linear-threshold learning algorithms on tasks involving superfluous attributes. In *Proceedings of the XII International conference on Machine Learning*, pages 353–361, 1995.

[Lit96] N. Littlestone. Mistake-driven bayes sports: Bounds for symmetric apobayesian learning algorithms. Technical report, NEC Research Institute, Princeton, NJ, 1996.

[LW94] N. Littlestone and M. K. Warmuth. The weighted majority algorithm. *Information and Computation*, 108:212–261, 1994.

[Vov90] Volodimir G. Vovk. Aggregating strategies. In *Proceedings of the 1990 Workshop on Computational Learning Theory*, pages 371–383, 1990.